# Automatic Generation of Social Tags for Music Recommendation

**Douglas Eck**[*]
Sun Labs, Sun Microsystems
Burlington, Mass, USA
`douglas.eck@umontreal.ca`

**Paul Lamere**
Sun Labs, Sun Microsystems
Burlington, Mass, USA
`paul.lamere@sun.com`

**Thierry Bertin-Mahieux**
Sun Labs, Sun Microsystems
Burlington, Mass, USA
`bertinmt@iro.umontreal.ca`

**Stephen Green**
Sun Labs, Sun Microsystems
Burlington, Mass, USA
`stephen.green@sun.com`

## Abstract

Social tags are user-generated keywords associated with some resource on the Web. In the case of music, social tags have become an important component of "Web2.0" recommender systems, allowing users to generate playlists based on use-dependent terms such as *chill* or *jogging* that have been applied to particular songs. In this paper, we propose a method for predicting these social tags directly from MP3 files. Using a set of boosted classifiers, we map audio features onto social tags collected from the Web. The resulting automatic tags (or *autotags*) furnish information about music that is otherwise untagged or poorly tagged, allowing for insertion of previously unheard music into a social recommender. This avoids the "cold-start problem" common in such systems. Autotags can also be used to smooth the tag space from which similarities and recommendations are made by providing a set of comparable baseline tags for all tracks in a recommender system.

## 1 Introduction

Social tags are a key part of "Web 2.0" technologies and have become an important source of information for recommendation. In the domain of music, Web sites such as Last.fm use social tags as a basis for recommending music to listeners. In this paper we propose a method for predicting social tags using audio feature extraction and supervised learning. These automatically-generated tags (or "autotags") can furnish information about music for which good, descriptive social tags are lacking. Using traditional information retrieval techniques a music recommender can use these autotags (combined with any available listener-applied tags) to predict artist or song similarity. The tags can also serve to smooth the tag space from which similarities and recommendations are made by providing a set of comparable baseline tags for all artists or songs in a recommender.

This is not the first attempt to predict something about textual data using music audio as input. Whitman & Rifkin [10], for example, provide an audio-driven model for predicting words found near artists in web queries . One main contribution of the work in this paper lies in the scale of our experiments. As is described in Section 4 we work with a social tag database of millions of tags applied to $\sim 100,000$ artists and an audio database of $\sim 90,000$ songs spanning many of the more popular of these artists. This compares favorably with previous attempts which by and large treat only very small datasets (e.g. [10] used 255 songs drawn from 51 artists.)

---

[*]Eck and Bertin-Mahieux currently at Dept. of Computer Science, Univ. of Montreal, Montreal , Canada

This paper is organized as follows: in Section 2 we describe social tags in more depth, including a description of how social tags can be used to avoid problems found in traditional collaborative filtering systems, as well as a description of the tag set we built for these experiments. In Section 3 we present an algorithm for autotagging songs based on labeled data collected from the Internet. In Section 4 we present experimental results and also discuss the ability to use model results for visualization. Finally, in Section 5 we describe our conclusions and future work.

## 2    Using social tags for recommendation

As the amount of online music grows, automatic music recommendation becomes an increasingly important tool for music listeners to find music that they will like. Automatic music recommenders commonly use collaborative filtering (CF) techniques to recommend music based on the listening behaviors of other music listeners. These CF recommenders (CFRs) harness the "wisdom of the crowds" to recommend music. Even though CFRs generate good recommendations there are still some problems with this approach. A significant issue for CFRs recommenders is the *cold-start* problem. A recommender needs a significant amount of data before it can generate good recommendations. For new music, music by an unknown artist with few listeners, a CFR cannot generate good recommendations. Another issue is the *lack of transparency* in recommendations [7]. A CFR cannot tell a listener why an artist was recommended beyond the description: "people who listen to X also listen to Y". Also, a CFR is relatively insensitive to multimodal uses of the same album or song. For example songs from an album (a single purchase in a standard CFR system) may be used in the context of dining, jogging and working. In each context, the reason the song was selected changes.

An alternative style of recommendation that addresses many of the shortcomings of a CFR is to recommend music based upon the similarity of "social tags" that have been applied to the music. Social tags are free text labels that music listeners apply to songs, albums or artists. Typically, users are motivated to tag as a way to organize their own personal music collection. The real strength of a tagging system is seen when the tags of many users are aggregated. When the tags created by thousands of different listeners are combined, a rich and complex view of the song or artist emerges. Table 1 show the top 21 tags and frequencies of tags applied to the band "The Shins". Users have applied tags associated with the genre (*Indie*, *Pop*, etc.), with the mood (*mellow*, *chill*), opinion (*favorite*, *love*), style (*singer-songwriter*) and context (*Garden State*). From these tags and their frequencies we learn much more about "The Shins" than we would from a traditional single genre assignment of "Indie Rock".

In this paper, we investigate the automatic generation of tags with properties similar to those generated by social taggers. Specifically, we introduce a machine learning algorithm that takes as input acoustic features and predicts social tags mined from the web (in our case, Last.fm). The model can then be used to tag new or otherwise untagged music, thus providing a partial solution to the cold-start problem.

For this research, we extracted tags and tag frequencies for nearly 100,000 artists from the social music website Last.fm using the Audioscrobbler web service [1]. The majority of tags describe audio content. Genre, mood and instrumentation account for 77% of the tags. See "extra material" for a breakdown of tag types.

Overcoming the cold-start problem is the primary motivation for this area of research. For new music or sparsely tagged music, we predict social tags directly from the audio and apply these automatically generated tags (called *autotags*) in lieu of traditionally applied social tags. By automatically tagging new music in this fashion, we can reduce or eliminate much of the cold-start problem.

## 3    An autotagging algorithm

We now describe a machine learning model which uses the *meta-learning* algorithm AdaBoost [5] to predict tags from acoustic features. This model is an extension of a previous model [3] which won the Genre Prediction Contest and was the 2nd place performer in the Artist Identification Contest at MIREX 2005 (ISMIR conference, London, 2005). The model has two principal advantages. First it selects features based on a feature's ability to minimize empirical error. We can therefore use the

| Tag | Freq | Tag | Freq | Tag | Freq |
|---|---|---|---|---|---|
| Indie | 2375 | The Shins | 190 | Punk | 49 |
| Indie rock | 1138 | Favorites | 138 | Chill | 45 |
| Indie pop | 841 | Emo | 113 | Singer-songwriter | 41 |
| Alternative | 653 | Mellow | 85 | Garden State | 39 |
| Rock | 512 | Folk | 85 | Favorite | 37 |
| Seen Live | 298 | Alternative rock | 83 | Electronic | 36 |
| Pop | 231 | Acoustic | 54 | Love | 35 |

Table 1: Top 21 tags applied to *The Shins*

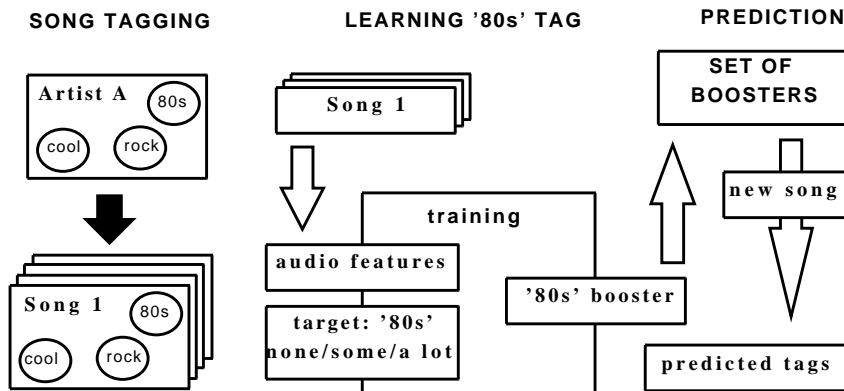

Figure 1: Overview of our model

model to eliminate useless feature sets by looking at the order in which those features are selected. We used this property of the model to discard many candidate features such as chromagrams (which map spectral energy onto the 12 notes of the Western musical scale) because the weak learners associated with those features were selected very late by AdaBoost. Second, though AdaBoost may need relatively more weak learners to achieve the same performance on a large dataset than a small one, the computation time for a single weak learner scales linearly with the number of training examples. Thus AdaBoost has the potential to scale well to very large datasets. Both of these properties are general to AdaBoost and are not explored further in this short paper. See [5, 9] for more.

## 3.1 Acoustic feature extraction

The features we use include 20 Mel-Frequency Cepstral Coefficients, 176 autocorrelation coefficients computed for lags spanning from 250msec to 2000msec at 10ms intervals, and 85 spectrogram coefficients sampled by constant-Q (or log-scaled) frequency (see [6] for descriptions of these standard acoustic features.)

The audio features described above are calculated over short windows of audio ( 100ms with 25ms overlap). This yields too many features per song for our purposes. To address this, we create "aggregate" features by computing individual means and standard deviations (i.e., independent Gaussians) of these features over 5s windows of feature data. When fixing hyperparameters for these experiments, we also tried a combination of 5s and 10s features, but saw no real improvement in results. For reasons of computational efficiency we used random sampling to retain a maximum of 12 aggregate features per song, corresponding to 1 minute of audio data.

## 3.2 Labels as a classification problem

Intuitively, automatic labeling would be a regression task where a learner would try to predict tag frequencies for artists or songs. However, because tags are sparse (many artist are not tagged at all; others like Radiohead are heavily tagged) this proves to be too difficult using our current Last.fm

dataset. Instead, we chose to treat the task as a classification one. Specifically, for each tag we try to predict if a particular artist has "none", "some" or "a lot" of a particular tag relative to other tags.

We normalize the tag frequencies for each artist so that artists having many tags can be compared to artists having few tags. Then for each tag, an individual artist is placed into a single class "none", "some" or "a lot" depending on the proportion of times the tag was assigned to that artist relative to other tags assigned to that artist. Thus if an artist received only 50 *rock* tags and nothing else, it would be treated as having "a lot" of rock. Conversely, if an artist received 5000 *rock* tags but 10,000 *jazz* tags it would be treated as having "some" *rock* and "a lot" of *jazz*. The specific boundaries between "none", "some" and "a lot" were decided by summing the normalized tag counts or all artists, generating a 100-bin histogram for each tag and moving the category boundaries such that an equal number of artists fall into each of the categories. In Figure 2 the histogram for "rock" is shown (with only 30 bins to make the plot easier to read). Note that most artists fall into the lowest bin (no or very few instances of the "rock" tag) and that otherwise most of the mass is in high bins. This was the trend for most tags and one of our motivations for using only 3 bins. As described in the paper we do not directly use the predictions of the "some" bin. Rather it serves as a class for holding those artists for which we cannot confidently say "none" or "a lot". See Figure 2 for an example.

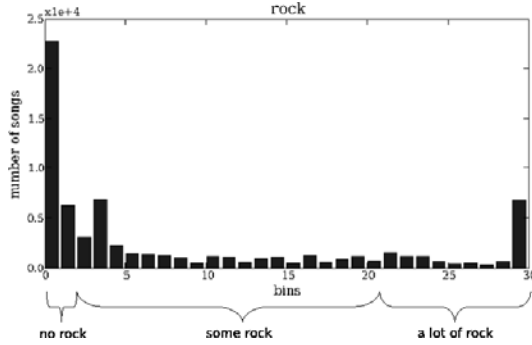

Figure 2: A 30-bin histogram of the proportion of "rock" tags to other tags for all songs in the dataset.

### 3.3  Tag prediction with AdaBoost

AdaBoost [5] is a *meta-learning* method that constructs a *strong classifier* from a set of simpler classifiers, called *weak learners* in an iterative way. Originally intended for binary classification, there exist several ways to extend it to multiclass classification. We use AdaBoost.MH [9] which treats multiclass classification as a set of one-versus-all binary classification problems. In each iteration $t$, the algorithm selects the best classifier, called $h^{(t)}$ from a pool of weak learners, based on its performance on the training set, and assigns it a coefficient $\alpha^{(t)}$. The input to the weak learner is a $d$-dimensional observation vector $x \in \Re^d$ containing audio features for one segment of aggregated data (5 seconds in our experiments). The output of $h^{(t)}$ is a binary vector $y \in \{-1, 1\}^k$ over the $k$ classes. $h_l^{(t)} = 1$ means a vote for class $l$ by a weak learner while $h^{(t)}, -1$ is a vote against. After $T$ iterations, the algorithm output is a vector-valued discriminant function:

$$g(x) = \sum_{t=1}^{T} \alpha^{(t)} h^{(y)}(x) \tag{1}$$

As weak learners we used single stumps, e.g. a binary threshold on one of the features. In previous work we also tried decision trees without any significant improvement. Usually we obtain a single label by taking the class with the most votes i.e $f(x) = \arg\max_l g_l(x)$, but in our model, we use the output value for each class rather than the argmax.

### 3.4  Generating autotags

For each aggregate segment, a booster yields a prediction over the classes "none", "some", and "a lot". A booster's raw output for a single segment might be (none:$-3.56$) (some:$0.14$) (a lot:$2.6$).

These segment predictions can then be combined to yield artist-level predictions. This can be achieved in two ways: a winning class can be chosen for each segment (in this example the class "a lot" would win with 2.6) and the mean over winners can be tallied for all segments belonging to an artist. Alternately we can skip choosing a winner and simply take the mean of the raw outputs for an artist's segments. Because we wanted to estimate tag frequencies using booster magnitude we used the latter strategy.

The next step is to transform these class for our individual social tag boosters into a bag of words to be associated with an artist. The most naive way to obtain a single value for *rock* is to look solely at the prediction for the "a lot" class. However this discards valuable information such as when a booster votes strongly "none". A better way to obtain a measure for *rock*-ness is to take the center of mass of the three values. However, because the values are not scaled well with respect to one another, we ended up with poorly scaled results. Another intuitive idea is simply to subtract the value of the "none" bin from the value of the "a lot" bin, the reasoning being that "none" is truly the opposite of "a lot". In our example, this would yield a *rock* strength of 7.16. In experiments for setting hyperparameters, this was shown to work better than other methods. Thus to generate our final measure of *rock*-ness, we ignore the middle bin ("some"). However this should not be taken to mean that the middle "some" bin is useless: the booster needed to learn to predict "some" during training thus forcing it to be more selective in predicting "none" and "a lot". As a large-margin classifier, AdaBoost tries to separate the classes as much as possible, so the magnitude of the values for each bin are not easily comparable. To remedy this, we normalize by taking the minimum and maximum prediction for each booster, which seems to work for finding similar artists. This normalization would not be necessary if we had good tagging data for all artists and could perform regression on the frequency of tag occurrence across artists.

## 4 Experiments

To test our model we selected the 60 most popular tags from the Last.fm crawl data described in Section 2. These tags included genres such as "Rock", "Electronica", and "Post Punk", mood-related terms such as "Chillout". The full list of tags and frequencies are available in the "extra materials". We collected MP3s for a subset of the artists obtained in our Audioscrobbler crawl. From those MP3s we extracted several popular acoustic features. In total our training and testing data included 89924 songs for 1277 artists and yielded more than 1 million 5s aggregate features.

### 4.1 Booster Errors

As described above, a classifier was trained to map audio features onto aggregate feature segments for each of the 60 tags. A third of the data was withheld for testing. Because each of the 60 boosters needed roughly 1 day to process, we did not perform cross-validation. However each booster was trained on a large amount of data relative to the number of decision stumps learned, making overfitting a remote possibility. Classification errors are shown in Table 2. These errors are broken down by tag in the annex for this paper. Using 3 bins and balanced classes, the random error is about 67%.

|  | Mean | Median | Min | Max |
|---|---|---|---|---|
| Segment | 40.93 | 43.1 | 21.3 | 49.6 |
| Song | 37.61 | 39.69 | 17.8 | 46.6 |

Table 2: Summary of test error (%) on predicting bins for songs and segments.

### 4.2 Evaluation measures

We use three measures to evaluate the performance of the model. The first *TopN* compares two ranked lists, a target "ground truth" list $A$ and our predicted list $B$. This measure is introduced in [2], and is intended to place emphasis on how well our list predicts the top few items of the target list. Let $k_j$ be the position in list $B$ of the $j$th element from list $A$. $\alpha_r = 0.5^{1/3}$, and $\alpha_c = 0.5^{2/3}$,

as in [2]. The result is a value between 0 (dissimilar) and 1 (identical top $N$),

$$s_i = \frac{\sum_{j=1}^{N} \alpha_r^j \alpha_c^{k_j}}{\sum_{l=1}^{N} (\alpha_r * \alpha_c)^l} \quad (2)$$

For the results produced below, we look at the top $N = 10$ elements in the lists.

Our second measure is Kendall's $Tau$, a classic measure in collaborative filtering which measures the number of discordant pairs in 2 lists. Let $R_A(i)$ be the rank of the element $i$ in list $A$, if $i$ is not explicitly present, $R_A(i) = length(A) + 1$. Let $C$ be the number of concordant pairs of elements $(i, j)$, e.g. $R_A(i) > R_A(j)$ and $R_B(i) < R_B(j)$. In a similar way, $D$ is the number of discordant pairs. We use $\tau$'s approximation in [8]. We also define $T_A$ and $T_B$ the number of ties in list $A$ and $B$. In our case, it's the number of pairs of artists that are in $A$ but not in $B$, because they end up having the same position $R_B = length(B) + 1$, and reciprocally. Kendall's tau value is defined as:

$$\tau = \frac{C - D}{sqrt((C + D + T_A)(C + D + T_B))} \quad (3)$$

Unless otherwise noted, we analyzed the top 50 predicted values for the target and predicted lists. Finally, we compute what we call the *TopBucket*, which is simply the percentage of common elements in the top $N$ of 2 ranked lists. Here as in *Kendall* we compare the top 50 predicted values unless otherwise noted.

## 4.3 Constructing ground truth

As has long been acknowledged [4] one of the biggest challenges in addressing this task is to find a reasonable "ground truth" against which to compare our results. We seek a similarity matrix among artists which is not overly biased by current popularity, and which is not built directly from the social tags we are using for learning targets. Furthermore we want to derive our measure using data that is freely available data on the web, thus ruling out commercial services such as AllMusic (www.allmusic.com). Our solution is to construct our ground truth similarity matrix using correlations from the listening habits of Last.fm users. If a significant number of users listen to artists $A$ and $B$ (regardless of the tags they may assign to that artist) we consider those two artists similar.

One challenge, of course, is that some users listen to more music than others and that some artists are more popular than others. Text search engines must deal with a similar problem: they want to ensure that frequently used words (e.g., *system*) do not outweigh infrequently used words (e.g., *prestidigitation*) and that long documents do not always outweigh short documents. Search engines assign a weight to each word in a document. The weight is meant to represent how important that word is for that document. Although many such weighting schemes have been described (see [11] for a comprehensive review), the most popular is the term frequency-inverse document frequency (or TF×IDF) weighting scheme. TF×IDF assigns high weights to words that occur frequently in a given document and infrequently in the rest of the collection. The fundamental idea is that words that are assigned high weights for a given document are good discriminators for that document from the rest of the collection. Typically, the weights associated with a document are treated as a vector that has its length normalized to one.

In the case of LastFM, we can consider an artist to be a "document", where the "words" of the document are the users that have listened to that artist. The TF×IDF weight for a given user for a given artist takes into account the global popularity of a given artist and ensures that users who have listened to more artists do not automatically dominate users who have listened to fewer artists. The resulting similarity measure seems to us to do a reasonable enough job of capturing artist similarity. Furthermore it does not seem to be overly biased towards popular bands. See "extra material" for some examples.

## 4.4 Similarity Results

One intuitive way to compare autotags and social tags is to look at how well the autotags reproduce the rank order of the social tags. We used the measures in Section 4.2 to measure this on 100 artists not used for training (Table 3). The results were well above random. For example, the top 5 autotags were in agreement with the top 5 social tags 61% of the time.

|          | TopN 10 | Kendall (N=5) | TopBucket (N=5) |
|----------|---------|---------------|-----------------|
| autotags | 0.636   | -0.099        | **61.0%**       |
| random   | 0.111   | -0.645        | 8.1%            |

Table 3: Results for all three measures on tag order for 100 out-of-sample artists.

A more realistic way to compare autotags and social tags is via their artist similarity predictions. We construct similarity matrices from our autotag results and from the Last.fm social tags used for training and testing. The similarity measure we used was *cosine similarity* $s_{cos}(A_1, A_2) = A_1 * A_2/(||A_1|| \, ||A_2||)$ where $A_1$ and $A_2$ are tag magnitudes for an artist. In keeping with our interest in developing a commercial system, we used all available data for generating the similarity matrices, including data used for training. (The chance of overfitting aside, it would be unwise to remove The Beatles from your recommender simply because you trained on some of their songs). The similarity matrix is then used to generate a ranked list of similar artists for each artist in the matrix. These lists are used to compute the measures describe in Section 4.2. Results are found at the top in Table 4.

One potential flaw in this experiment is that the ground truth comes from the same data source as the training data. Though the ground truth is based on user listening counts and our learning data comes from aggregate tagging counts, there is still a clear chance of contamination. To investigate this, we selected the autotags and social tags for 95 of the artists from the USPOP database [2]. We constructed a ground truth matrix based on the 2002 MusicSeer web survey eliciting similarity rankings between artists from appro 1000 listeners [2]. These results show much closer correspondence between our autotag results and the social tags from Last.fm than the previous test. See bottom, Table 4.

| Groundtruth | Model       | TopN 10 | Kendall 50 | TopBucket 20 |
|-------------|-------------|---------|------------|--------------|
| Last.FM     | social tags | 0.26    | -0.23      | 34.6%        |
|             | autotags    | 0.118   | -0.406     | 22.5%        |
|             | random      | 0.005   | -0.635     | 3.9%         |
| MusicSeer   | social tags | 0.237   | -0.182     | 29.7%        |
|             | autotags    | 0.184   | -0.161     | 28.2%        |
|             | random      | 0.051   | -0.224     | 21.5%        |

Table 4: Performance against Last.Fm (top) and MusicSeer (bottom) ground truth.

It is clear from these previous two experiments that our autotag results do not outperform the social tags on which they were trained. Thus we asked whether combining the predictions of the autotags with the social tags would yield better performance than either of them alone. To test this we blended the autotag similarity matrix $S_a$ with the social tag matrix $S_s$ using $\alpha S_a + (1 - \alpha)S_s$. The results shown in Figure 3 show a consistent performance increase when blending the two similarity sources. It seems clear from these results that the autotags are of value. Though they do not outperform the social tags on which they were trained, they do yield improved performance when combined with social tags. At the same time they are driven entirely by audio and so can be applied to new, untagged music. With only 60 tags the model makes some reasonable predictions. When more boosters are trained, it is safe to assume that the model will perform better.

## 5   Conclusion and future work

The work presented here is preliminary, but we believe that a supervised learning approach to autotagging has substantial merit. Our next step is to compare the performance of our boosted model to other approaches such as SVMs and neural networks. The dataset used for these experiments is already larger than those used for published results for genre and artist classification. However, a dataset another order of magnitude larger is necessary to approximate even a small commercial database of music. A further next step is comparing the performance of our audio features with other sets of audio features.

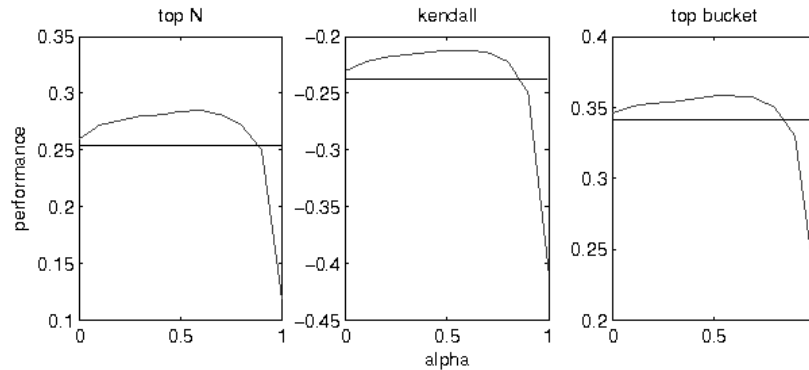

Figure 3: Similarity performance results when autotag similarities are blended with social tag similarities. The horizontal line is the performance of the social tags against ground truth.

We plan to extend our system to predict many more tags than the current set of 60 tags. We expect the accuracy of our system to improve as we extend our tag set, especially as we add tags such as *Classical* and *Folk* that are associated with whole genres of music. We will also continue exploring ways in which the autotag results can drive music visualization. See "extra examples" for some preliminary work.

Our current method of evaluating our system is biased to favor popular artists. In the future, we plan to extend our evaluation to include comparisons with music similarity derived from human analysis of music. This type of evaluation should be free of popularity bias. Most importantly, the machine-generated autotags need to be tested in a social recommender. It is only in such a context that we can explore whether autotags, when blended with real social tags, will in fact yield improved recommendations.

## References

[1] Audioscrobbler. Web Services described at http://www.audioscrobbler.net/data/webservices/.

[2] A. Berenzweig, B. Logan, D. Ellis, and B. Whitman. A large-scale evaluation of acoustic and subjective music similarity measures. In *Proceedings of the 4th International Conference on Music Information Retrieval (ISMIR 2003)*, 2003.

[3] J. Bergstra, N. Casagrande, D. Erhan, D. Eck, and B. Kégl. Aggregate features and AdaBoost for music classification. *Machine Learning*, 65(2-3):473–484, 2006.

[4] D. Ellis, B. Whitman, A. Berenzweig, and S. Lawrence. The quest for ground truth in musical artist similarity. In *Proceedings of the 3th International Conference on Music Information Retrieval (ISMIR 2002)*, 2002.

[5] Y. Freund and R.E. Shapire. Experiments with a new boosting algorithm. In *Machine Learning: Proceedings of the Thirteenth International Conference*, pages 148–156, 1996.

[6] B. Gold and N. Morgan. *Speech and Audio Signal Processing: Processing and Perception of Speech and Music*. Wiley, Berkeley, California., 2000.

[7] Jonathan L. Herlocker, Joseph A. Konstan, and John Riedl. Explaining collaborative filtering recommendations. In *Computer Supported Cooperative Work*, pages 241–250, 2000.

[8] Jonathan L. Herlocker, Joseph A. Konstan, Loren G. Terveen, and John T. Riedl. Evaluating collaborative filtering recommender systems. *ACM Trans. Inf. Syst.*, 22(1):5–53, 2004.

[9] R. E. Schapire and Y. Singer. Improved boosting algorithms using confidence-rated predictions. *Machine Learning*, 37(3):297–336, 1999.

[10] Brian Whitman and Ryan M. Rifkin. Musical query-by-description as a multiclass learning problem. In *IEEE Workshop on Multimedia Signal Processing*, pages 153–156. IEEE Signal Processing Society, 2002.

[11] Justin Zobel and Alistair Moffat. Exploring the similarity space. *SIGIR Forum*, 32(1):18–34, 1998.

